# A feature selection algorithm based on the global minimization of a generalization error bound

**Dori Peleg**
Department of Electrical Engineering
Technion
Haifa, Israel
dorip@tx.technion.ac.il

**Ron Meir**
Department of Electrical Engineering
Technion
Haifa, Israel
rmeir@tx.technion.ac.il

## Abstract

A novel linear feature selection algorithm is presented based on the global minimization of a data-dependent generalization error bound. Feature selection and scaling algorithms often lead to non-convex optimization problems, which in many previous approaches were addressed through gradient descent procedures that can only guarantee convergence to a local minimum. We propose an alternative approach, whereby the global solution of the non-convex optimization problem is derived via an equivalent optimization problem. Moreover, the convex optimization task is reduced to a conic quadratic programming problem for which efficient solvers are available. Highly competitive numerical results on both artificial and real-world data sets are reported.

## 1 Introduction

This paper presents a new approach to feature selection for linear classification where the goal is to learn a decision rule from a training set of pairs $S_n = \left\{x^{(i)}, y^{(i)}\right\}_{i=1}^{n}$, where $x^{(i)} \in \mathbb{R}^d$ are input patterns and $y^{(i)} \in \{-1, 1\}$ are the corresponding labels. The goal of a classification algorithm is to find a separating function $f(\cdot)$, based on the training set, which will generalize well, i.e. classify new patterns with as few errors as possible. Feature selection schemes often utilize, either explicitly or implicitly, scaling variables, $\{\sigma_j\}_{j=1}^{d}$, which multiply each feature. The aim of such schemes is to optimize an objective function over $\sigma \in \mathbb{R}^d$. Feature selection can be viewed as the case $\sigma_j \in \{0, 1\}$, $j = 1, \ldots, d$, where a feature $j$ is removed if $\sigma_j = 0$. The more general case of feature scaling is considered here, i.e. $\sigma_j \in \mathbb{R}_+$. Clearly feature selection is a special case of feature scaling.

The overwhelming majority of feature selection algorithms in the literature, separate the feature selection and classification tasks, while solving either a combinatorial or a non-convex optimization problem (e.g. [1],[2],[3],[4]). In either case there is no guarantee of efficiently locating a global optimum. This is particularly problematic in large scale classification tasks which may initially contain several thousand features. Moreover, the objective function of many feature selection algorithms is unrelated to the Generalization Error (GE). Even for global solutions of such algorithms there is no theoretical guarantee of proximity to the minimum of the GE.

To overcome the above shortcomings we propose a feature selection algorithm based on the Global Minimization of an Error Bound (GMEB). This approach is based on simultaneously finding the optimal classifier and scaling factors of each feature by minimizing a GE bound. As in previous feature selection algorithms, a non-convex optimization problem must be solved. A novelty of this paper is the use of the *equivalent optimization problems* concept, whereby a global optimum is guaranteed in polynomial time.

The development of the GMEB algorithm begins with the design of a GE bound for feature selection. This is followed by formulating an optimization problem which minimizes this bound. Invariably, the resulting problem is non-convex. To avoid the drawbacks of solving non-convex optimization problems, an equivalent convex optimization problem is formulated whereby the *exact* global optimum of the non-convex problem can be computed. Next the dual problem is derived and formulated as a Conic Quadratic Programming (CQP) problem. This is advantageous because efficient CQP algorithms are available. Comparative numerical results on both artificial and real-world datsets are reported.

The notation and definitions were adopted from [5]. All vectors are column vectors unless transposed. Mathematical operators on scalars such as the square root are expanded to vectors by operating componentwise. The notation $\mathbb{R}_+$ denotes nonnegative real numbers. The notation $x \preceq y$ denotes componentwise inequality between vectors $x$ and $y$ respectively. A vector with all components equal to one is denoted as $\mathbf{1}$. The domain of a function $f$ is denoted as **dom** $f$. The set of points for which the objective and all the constraint functions are defined is called the *domain* of the optimization problem, $\mathcal{D}$. For lack of space, only proof sketches will be presented; the complete proofs are deferred to the full paper.

## 2   The Generalization Error Bounds

We establish GE bounds which are used to motivate an effective algorithm for feature scaling. Consider a sample $S_n = \{(x^{(1)}, y^{(1)}), \ldots, (x^{(n)}, y^{(n)})\}$, $x^{(i)} \in \mathcal{X} \subseteq \mathbb{R}^d$, $y^{(i)} \in \mathcal{Y}$, where $(x^{(i)}, y^{(i)})$ are generated independently from some distribution $P$. A set of nonnegative variables $\sigma = (\sigma_1, \ldots, \sigma_d)^T$ is introduced to allow the additional freedom of feature scaling. The scaling variables $\sigma$ transform the linear classifiers from $f(x) = w^T x + b$ to $f(x) = w^T \Sigma x + b$, where $\Sigma = \mathbf{diag}(\sigma)$. It may seem at first glance that these classifiers are essentially the same since $w$ can be redefined as $\Sigma w$. However the role of $\sigma$ is to offer an extra degree of freedom to scale the features *independently* of $w$, in a way which can be exploited by an optimization algorithm.

For a real-valued classifier $f$, the $0 - 1$ loss is the probability of error given by $P(yf(x) \le 0) = \mathbf{E}I(yf(x) \le 0)$, where $I(\cdot)$ is the indicator function.

**Definition 1** *The margin cost function $\phi_\gamma : \mathbb{R} \to \mathbb{R}_+$ is defined as $\phi_\gamma(z) = 1 - z/\gamma$ if $z \le \gamma$, and zero otherwise (note that $I(yf(x) \le 0) \le \phi_\gamma(yf(x))$).*

Consider a classifier $f$ for which the input features have been rescaled, namely $f(\Sigma x)$ is used instead of $f(x)$. Let $\mathcal{F}$ be some class of functions and let $\hat{\mathbf{E}}_n$ be the empirical mean. Using standard GE bounds, one can establish that for any choice of $\sigma$, with probability at least $1 - \delta$, for any $f \in \mathcal{F}$

$$P(yf(\Sigma x) \le 0) \le \hat{\mathbf{E}}_n \phi_\gamma(yf(\Sigma x)) + \Omega(f, \delta, \sigma), \qquad (1)$$

for some appropriate complexity measure $\Omega$ depending on the bounding technique.

Unfortunately, (1) cannot be used directly when attempting to select optimal values of the variables $\sigma$ because the bound is not *uniform* in $\sigma$. In particular, we need a result which holds with probability $1 - \delta$ for *every* choice of $\sigma$.

**Definition 2** *The indices of training patterns with labels $\{-1, 1\}$ are denoted by $I_-, I_+$ respectively. The cardinalities of the sets $I_-, I_+$ are $n_-, n_+$ respectively. The empirical mean of the second order moment of the jth feature over the training patterns belonging to indices $I_-, I_+$ are $v_j^- = \frac{1}{n_-} \sum_{i \in I_-} \left( x_j^{(i)} \right)^2$, $v_j^+ = \frac{1}{n_+} \sum_{i \in I_+} \left( x_j^{(i)} \right)^2$ respectively.*

**Theorem 3** *Fix $B, r, \gamma > 0$, and suppose that $\{(x^{(i)}, y^{(i)})\}_{i=1}^n$ are chosen independently at random according to some probability distribution $P$ on $\mathcal{X} \times \{\pm 1\}$, where $\|x\| \leq r$ for $x \in \mathcal{X}$. Define the class of functions $\mathcal{F}$*

$$\mathcal{F} = \left\{ f : f(x) = w^T \Sigma x + b, \; \|w\| \leq B, \; |b| \leq r, \; \sigma \succeq \mathbf{0} \right\}.$$

*Let $\sigma_0$ be an arbitrary positive number, and set $\grave{\sigma}_j = 2 \max(\sigma_j, \sigma_0)$. Then with probability at least $1 - \delta$, for every function $f \in \mathcal{F}$*

$$P\left(yf(x) \leq 0\right) \leq \hat{\mathbf{E}}_n \phi_\gamma \left(yf(x)\right) + \frac{2B}{\gamma} \left( \frac{\sqrt{n_+}}{n} \sqrt{\sum_{j=1}^d v_j^+ \grave{\sigma}_j^2} + \frac{\sqrt{n_-}}{n} \sqrt{\sum_{j=1}^d v_j^- \grave{\sigma}_j^2} \right) + \Lambda,$$

(2)

*where $K(\sigma) = (B\|\grave{\sigma}\| + 1)r$ and $\Lambda = \frac{\Lambda(\sigma, \gamma, \delta)}{\sqrt{n}}$,*

$$\Lambda(\sigma, \gamma, \delta) = \frac{2r}{\gamma} + K(\sigma) \sqrt{2 \sum_{j=1}^d \ln \log_2 \frac{\grave{\sigma}_j}{\sigma_0}} + K(\sigma) \left( \frac{2}{\gamma} + 1 \right) \sqrt{2 \ln \frac{2}{\delta}}.$$

**Proof sketch** We begin by assuming a *fixed* upper bound on the values of $\sigma_j$, say $\sigma_j \leq s_j$, $j = 1, 2, \ldots, d$. This allows us to use the methods developed in [6] in order to establish upper bounds on the Rademacher complexity of the class $\mathcal{F}$, where $\sigma_j \leq s_j$ for all $j$. Finally, a simple variant of the union bound (the so-called multiple testing lemma) is used in order to obtain a bound which is uniform with respect to $\sigma$ (see the proof technique of Theorem 10 in [6]).

In principle, we would like to minimize the r.h.s. of (2) with respect to the variables $w, \sigma, b$. However, in this work the focus is only on the *data-dependent* terms in (2), which include the empirical error term and the weighted norms of $\sigma$. Note that all other terms of (2) are of the same order of magnitude (as a function of $n$), but do not depend explicitly on the data. It should be commented that the extra terms appearing in the bound arise because of the assumed unboundedness of $\sigma$. Assuming $\sigma$ to be bounded, e.g. $\sigma \preceq s$, as is the case in most other bounds in the literature, one may replace $\sigma$ by $s$ in all terms except the first two, thus removing the explicit dependence on $\sigma$.

The data-dependent terms of the GE bound (2) are the basis of the objective function

$$\frac{1}{n\gamma} \sum_{i=1}^n \phi_\gamma \left( y^{(i)} f(x^{(i)}) \right) + \frac{C_+ \sqrt{n_+}}{n\gamma} \sqrt{\sum_{j=1}^d v_j^+ \sigma_j^2} + \frac{C_- \sqrt{n_-}}{n\gamma} \sqrt{\sum_{j=1}^d v_j^- \sigma_j^2}, \quad (3)$$

where $C_+ = C_- = 4$ and the variables are subject to $w^T w \leq 1$, $\sigma \succeq 0$. The transition was performed by setting $B = 1$, and replacing $\grave{\sigma}$ by $2\sigma$ (assuming that $\sigma > \sigma_0$).

Due to the fact that only the sign of $f$ determines the estimated labels, it can be multiplied by any positive factor and produce identical results. The constraint on the norm of $w$ induces a normalization on the classifier $f(x) = w^T x + b$, without which the classifier is not unique. However, by introducing the scale variables $\sigma$, the classifier was transformed to $f(x) = w^T \Sigma x + b$. Hence, despite the constraint on $w$, the classifier is not unique again. If the variable $\gamma$ in (3) is set to an arbitrary positive constant then the solution is unique. This is true because $\gamma$ appears in (3) only in the expressions $\frac{b}{\gamma}, \frac{\sigma_1}{\gamma}, \ldots, \frac{\sigma_d}{\gamma}$. We chose $\gamma = 1$.

The objective function is comprised of two elements: (1) the mean of the penalty on the training errors (2) and two weighted $l_2$ norms of the scale variables $\sigma$. The second term acts as the feature selection element. Note that the values of $C_+, C_-$ following from Theorem 3 depend specifically on the bounding technique used in the proof. To allow more generality and flexibility in practical applications, we propose to turn the norm terms of (3) into inequality constraints which are bounded by hyperparameters $R_+, R_-$ respectively. The interpretation of these hyperparameters is essentially the number of informative features. We propose that $R_+, R_-$ are chosen via a Cross Validation (CV) scheme. These hyperparameters enable fine-tuning a general classifier to a specific classification task as is done in many other classification algorithms such as the SVM algorithm.

Note that the present bound is sensitive to a shift of the features. Therefore, as a preprocessing step the features of the training patterns should be set to zero mean and the features of the test set shifted accordingly.

## 3   The primal non-convex optimization problem

The problem of minimizing (3) with $\gamma = 1$ can then be expressed as

$$
\begin{aligned}
&\text{minimize} \quad \mathbf{1}^T \xi \\
&\text{subject to} \quad w^T w \le 1 \\
&\qquad\qquad y^{(i)}(\textstyle\sum_{j=1}^d x_j^{(i)} w_j \sigma_j + b) \ge 1 - \xi_i, \ \ i = 1, \dots, n \\
&\qquad\qquad R_+ \ge \textstyle\sum_{j=1}^d v_j^+ \sigma_j^2 \\
&\qquad\qquad R_- \ge \textstyle\sum_{j=1}^d v_j^- \sigma_j^2 \\
&\qquad\qquad \xi, \sigma \succeq 0,
\end{aligned}
\tag{4}
$$

with variables $w, \sigma \in \mathbb{R}^d, \xi \in \mathbb{R}^n, b \in \mathbb{R}$. Note that the constant value $\frac{1}{n}$ was discarded.

**Remark 4** *Consider a solution of problem* (4) *in which* $\sigma_j^\star = 0$ *for some feature* $j$*. Only the constraint* $w^T w \le 1$ *affects the value of* $w_j^\star$*. A unique solution is established by setting* $\sigma_j^\star = 0 \Rightarrow w_j^\star = 0$*. If the original solution* $w^\star$ *satisfies the constraint* $w^T w \le 1$ *then the amended solution will also satisfy the constraint and won't affect the value of the objective function.*

The functions $w_j \sigma_j$ in the second inequality constraints are neither convex nor concave (in fact they are quasiconcave [5]). To make matters worse, the functions $w_j \sigma_j$ are multiplied by constants $-y^{(i)} x_j^{(i)}$ which can be either positive or negative. Consequently problem (4) is *not* a convex optimization problem. The objective of Section 3.1 is to find the global minimum of (4) in polynomial time despite its non-convexity.

### 3.1   Convexification

In this paper the informal definition of equivalent optimization problems is adopted from [5, pp. 130–135]: two optimization problems are called *equivalent* if from a solution of one, a solution of the other is found, and vice versa. Instead of detailing a complicated formal definition of general equivalence, the specific equivalence relationships utilized in this paper are either formally introduced or cited from [5].

The functions $w_j \sigma_j$ in problem (4) are not convex and the signs of the multiplying constants $-y^{(i)} x_j^{(i)}$ are data dependant. The only functions that remain convex irrespective of the sign of the constants which multiply them are linear functions. Therefore the functions $w_j \sigma_j$ must be transformed into linear functions.

However, such a transformation must also maintain the convexity of the objective function and the remaining constraints. For this purpose the *change of variables* equivalence relationship, described in appendix A, was utilized. The transformation $\phi : \mathbb{R}^d \times \mathbb{R}^d \to \mathbb{R}^d \times \mathbb{R}^d$ was used on the variables $w, \sigma$:

$$\sigma_j = +\sqrt{\tilde{\sigma}_j}, \ w_j = \frac{\tilde{w}_j}{\sqrt{\tilde{\sigma}_j}}, \quad j = 1, \ldots, d, \tag{5}$$

where $\mathbf{dom}\,\phi = \{(\tilde{\sigma}, \tilde{w}) | \tilde{\sigma} \succeq 0\}$. If $\tilde{\sigma}_j = 0$ then $\sigma_j = w_j = 0$ without regard to the value of $\tilde{w}_j$, in accordance with remark 4. Transformation (5) is clearly one-to-one and $\phi(\mathbf{dom}\,\phi) \supseteq \mathcal{D}$.

**Lemma 5** *The problem*

$$
\begin{aligned}
\text{minimize} \quad & \mathbf{1}^T \xi \\
\text{subject to} \quad & y^{(i)}(\tilde{w}^T x^{(i)} + b) \geq 1 - \xi_i, \ i = 1, \ldots, n \\
& \sum_{j=1}^d \frac{\tilde{w}_j^2}{\tilde{\sigma}_j} \leq 1 \\
& R_+ \geq (v^+)^T \tilde{\sigma} \\
& R_- \geq (v^-)^T \tilde{\sigma} \\
& \xi, \tilde{\sigma} \succeq 0
\end{aligned}
\tag{6}
$$

*is convex and equivalent to the primal non-convex problem* (4) *with transformation* (5).

Note that since $\tilde{w}_j = w_j \sigma_j$, the new classifier is $f(x) = \tilde{w}^T x + b$. Therefore there is no need to use transformation (5) to obtain the desired classifier. Also one can use Schur's complement [5] to transform the non-linear constraint into a sparse linear matrix inequality constraint

$$\begin{bmatrix} \Sigma & w \\ w^T & 1 \end{bmatrix} \succeq 0.$$

Thus problem (6) can be cast as a Semi-Definite Programming (SDP) problem. The primal problem therefore, consists of $n + 2d + 1$ variables, $2n + d + 2$ linear inequality constraints and a linear matrix inequality of $[(d+1) \times (d+1)]$ dimensions. Although the primal problem (6) is convex, it heavily relies on the number of features $d$ which is typically the bottleneck for feature selection datasets. To alleviate this dependency the Dual problem is formulated.

**Theorem 6 (Dual problem)** *The dual optimization problem associated with problem* (6) *is*

$$
\begin{aligned}
\text{maximize} \quad & \mathbf{1}^T \mu - \mu_1 - R_+ \mu_+ - R_- \mu_- \\
\text{subject to} \quad & \left( \sum_{i=1}^n \mu_i y^{(i)} x_j^{(i)}, 2\mu_1, (\mu_+ v_j^+ + \mu_- v_j^-) \right) \in K^r \quad , j = 1, \ldots, d \\
& \mu^T y = 0 \\
& 0 \preceq \mu \preceq \mathbf{1} \\
& \mu_+, \mu_- \geq 0,
\end{aligned}
\tag{7}
$$

*where $K^r$ is the Rotated Quadratic Cone (RQC) $K^r = \{(x, y, z) \in \mathbb{R}^n \times \mathbb{R} \times \mathbb{R} | x^T x \leq 2yz, y \geq 0, z \geq 0\}$ and with the variables $\mu \in \mathbb{R}^n, \mu_1, \mu_2 \in \mathbb{R}$.*

**Theorem 7 (Strong duality)** *Strong duality holds between problems* (6) *and* (7).

The dual problem (7) is a CQP problem. The number of variables is $n + 3$, there are $2n + 2$ linear inequality constraints, a single linear equality constraint and $d$ RQC inequality constraints. Due to the reduced computational complexity we used the dual formulation in all the experiments.

# 4 Experiments

Several algorithms were comparatively evaluated on a number of artificial and real world two class problem datasets. The GMEB algorithm was compared to the linear SVM (standard SVM with linear kernel) and the $l_1$ SVM classifier [7].

## 4.1 Experimental Methodology

The algorithms are compared by two criteria: the number of selected features and the error rates. The weight assigned by a linear classifier to a feature $j$, determines whether it shall be 'selected' or 'rejected'. This weight must fulfil at least one of the following two requirements:

1. Absolute measure - $|w_j| \geq \epsilon$.

2. Relative measure - $\frac{|w_j|}{\max_j\{|w_j|\}} \geq \epsilon$.

In this paper $\epsilon = 0.01$ was used. Ideally, $\epsilon$ should be set adaptively. Note that for the GMEB algorithm $\tilde{w}$ should be used.

The definition of the error rate is intrinsically entwined with the protocol for determining the hyperparameter. Given an *a-priori* partitioning of the dataset into training and test sets, the following protocol for determining the value of $R_+, R_-$ and defining the error rate is suggested:

1. Define a set $\mathcal{R}$ of values of the hyperparameters $R_+, R_-$ for all datasets. The set $\mathcal{R}$ consists of a predetermined number of values. For each algorithm the cardinality $|\mathcal{R}| = 49$ was used.
2. Calculate the N-fold CV error for each value of $R_+, R_-$ from set $\mathcal{R}$ on the *training* set. Five fold CV was used throughout all the datasets.
3. Use the classifier with the value of $R_+, R_-$ which produced the lowest CV error to classify the test set. This is the reported error rate.

If the dataset is not partitioned *a-priori* into a training and test set, it is randomly divided into $n_p$ contiguous training and 'test' sets. Each training set contains $n\frac{n_p-1}{n_p}$ patterns and the corresponding test set consists of $\frac{n}{n_p}$ patterns. Once the dataset is thus partitioned, the above steps $1-3$ can be implemented. The error rate and the number of selected features are then defined as the average on the $n_p$ problems. The value $n_p = 10$ was used for all datasets, where an *a-priori* partitioning was not available.

The hyperparameter sets $\mathcal{R}$ used for the GMEB algorithm consisted of $7 \times 7$ linearly spaced values between 1 and 10. For the SVM algorithms the set $\mathcal{R}$ consisted of the values $\frac{\Lambda}{1-\Lambda}$ where $\Lambda = \{0.02, 0.04, \ldots, 0.98\}$, i.e. 49 linearly spaced values between 0.02 and 0.98.

## 4.2 Data sets

Tests were performed on the 'Linear problem' synthetic datasets as described in [2], and eight real-world problems. The number of features, the number of patterns and the partitioning into train and test sets of the real-world datasets are detailed in Table 2. The datasets were taken form the UCI repository unless stated otherwise. Dataset (1) is termed Wisconsin Diagnostic Breast Cancer 'WDBC', (2) 'Multiple Features' dataset, which was first introduced by ([8]), (3) the 'Internet Advertisements' dataset, was separated into a training and test set randomly, (4) the 'Colon' dataset, taken from ([2]), (5) the 'BUPA' dataset, (6) the 'Pima Indians Diabetes' dataset, (7) the 'Cleveland heart disease' dataset, and (8), the 'Ionosphere' dataset.

Table 1: Mean and standard deviation of the mean of test error rate percentage on synthetic datasets given $n$ training patterns. The number of selected features is in brackets.

| $n$ | SVM | $l_1$ SVM | GMEB |
|---|---|---|---|
| 10 | $46.2 \pm 1.9$ ($197.1\pm2.1$) | $49.6 \pm 1.9$ ($77.7\pm83.8$) | $\mathbf{33.8 \pm 14.2}$ ($\mathbf{3.7\pm2.1}$) |
| 20 | $44.9 \pm 2.1$ ($196.8\pm1.9$) | $38.5 \pm 12.7$ ($10.7\pm6.1$) | $\mathbf{13.9 \pm 7.2}$ ($\mathbf{4.8\pm2.7}$) |
| 30 | $43.6 \pm 1.7$ ($196.7\pm2.8$) | $27.4 \pm 12.4$ ($14.5\pm8.7$) | $\mathbf{7.1 \pm 5.6}$ ($\mathbf{5.1\pm2.3}$) |
| 40 | $41.8 \pm 1.9$ ($197.2\pm1.8$) | $19.2 \pm 6.9$ ($16.2\pm11.1$) | $\mathbf{5.0 \pm 3.5}$ ($\mathbf{5.5\pm2.1}$) |
| 50 | $41.9 \pm 1.8$ ($196.6\pm2.6$) | $16.0 \pm 5.3$ ($18.4\pm11.3$) | $\mathbf{3.1 \pm 2.7}$ ($\mathbf{5.1\pm1.8}$) |

Table 2: The real-world datasets and the performance of the algorithms. The set $\mathcal{R}$ for the linear SVM algorithm and for datasets 1,5,6 had to be set to $\Lambda$ to allow convergence.

| Feat. | Patt. | Linear SVM | $l_1$ SVM | GMEB |
|---|---|---|---|---|
| 30 | 569 | $5.3\pm0.8$ ($27.3\pm0.3$) | $4.9\pm1.1$ ($16.4\pm1.3$) | $\mathbf{4.2\pm0.9}$ ($\mathbf{6.0\pm0.3}$) |
| 649 | 200/1800 | $0.3$ ($616$) | $3.5$ ($\mathbf{15}$) | $\mathbf{0.2}$ ($32$) |
| 1558 | 200/3080 | $5.3$ ($322$) | $\mathbf{4.7}$ ($\mathbf{12}$) | $5.5$ ($98$) |
| 2000 | 62 | $13.6\pm5.9$ ($1941.8\pm1.9$) | $\mathbf{10.7\pm4.4}$ ($\mathbf{23.3\pm1.5}$) | $\mathbf{10.7\pm4.4}$ ($59.1\pm25.0$) |
| 6 | 345 | $\mathbf{33.1\pm3.5}$ ($6.0\pm0.0$) | $33.6\pm3.6$ ($5.9\pm0.1$) | $34.2\pm4.4$ ($\mathbf{5.4\pm0.5}$) |
| 8 | 768 | $22.8\pm1.5$ ($5.8\pm0.2$) | $22.9\pm1.4$ ($5.8\pm0.2$) | $\mathbf{22.5\pm1.8}$ ($\mathbf{4.8\pm0.2}$) |
| 13 | 297 | $17.5\pm1.9$ ($11.6\pm0.2$) | $16.8\pm1.6$ ($10.7\pm0.3$) | $\mathbf{15.5\pm2.0}$ ($9.1\pm0.3$) |
| 34 | 351 | $11.7\pm2.6$ ($32.8\pm0.2$) | $12.0\pm2.3$ ($27.9\pm1.6$) | $\mathbf{10.0\pm2.3}$ ($\mathbf{12.1\pm1.7}$) |

## 4.3  Experimental results

Table 1 provides a comparison of the GMEB algorithm with the SVM algorithms on the synthetic datasets. The Bayes error is $0.4\%$. For further numerical comparison see [3]. Note that the number of features selected by the $l_1$ SVM and the GMEB algorithms increase with the sample size. A possible explanation for this observation is that with only a few training patterns a small training error can be achieved by many subsets containing a small number of features, i.e. a sparse solution. The particular subset selected is essentially random, leading to a large test error, possibly due to overfitting.

For all the synthetic datasets the GMEB algorithm clearly attained the lowest error rates. On the real-world datasets it produced the lowest error rates and the smallest number of features for the majority of datasets investigated.

## 4.4  Discussion

The GMEB algorithm performs comparatively well against the linear and $l_1$ SVM algorithms, in regard to both the test error and the number of selected features. A possible explanation is that the $l_1$ SVM algorithm performs both classification and feature selection with the same variable $w$. In contrast, the GMEB algorithm performs the feature selection and classification simultaneously, while using variables $\sigma$ and $w$ respectively. The use of two variables also allows the GMEB algorithm to reduce the weight of a feature $j$ with both $w_j$ and $\sigma_j$, while the $l_1$ SVM uses only $w_j$. Perhaps this property of GMEB could explain why it produces comparable (and at times better) results than the SVM algorithms both in classification problems where feature selection is and is not required.

## 5 Summary and future work

This paper presented a feature selection algorithm motivated by minimizing a GE bound. The *global* optimum of the objective function is found by solving a non-convex optimization problem. The equivalent optimization problems technique reduces this task to a convex problem. The dual problem formulation depends more weakly on the number of features $d$ and this enabled an extension of the GMEB algorithm to large scale classification problems.

The GMEB classifier is a linear classifier. Linear classifiers are the most important type of classifiers in a feature selection framework because feature selection is highly susceptible to overfitting. We believe that the GMEB algorithm is just the first of a series of algorithms which may globally minimize increasingly tighter bounds on the generalization error.

**Acknowledgment** R.M. is partially supported by the fund for promotion of research at the Technion and by the Ollendorff foundation of the Electrical Engineering department at the Technion.

## A Change of variables

Consider optimization problem

$$\begin{aligned} \text{minimize} \quad & f_0(x) \\ \text{subject to} \quad & f_i(x) \le 0, \quad i = 1, \dots, m. \end{aligned} \tag{8}$$

Suppose $\phi : \mathbb{R}^n \to \mathbb{R}^n$ is one-to-one, with image covering the problem domain $\mathcal{D}$, i.e., $\phi(\mathbf{dom}\,\phi) \supseteq \mathcal{D}$. We define functions $\tilde{f}_i$ as $\tilde{f}_i(z) = f_i(\phi(z)), i = 0, \dots, m$. Now consider the problem

$$\begin{aligned} \text{minimize} \quad & \tilde{f}_0(z) \\ \text{subject to} \quad & \tilde{f}_i(z) \le 0, \quad i = 1, \dots, m, \end{aligned} \tag{9}$$

with variable $z$. Problem (8) and (9) are said to be related by the change of variable $x = \phi(z)$ and are equivalent: if $x$ solves the problem (8), then $z = \phi^{-1}(x)$ solves problem(9); if $z$ solves problem (9), then $x = \phi(z)$ solves problem (8).

## References

[1] Y. Grandvalet and S. Canu. Adaptive scaling for feature selection in svms. In S. Thrun S. Becker and K. Obermayer, editors, *Advances in Neural Information Processing Systems 15*, pages 553–560. MIT Press, 2003.

[2] Jason Weston, Sayan Mukherjee, Olivier Chapelle, Massimiliano Pontil, Tomaso Poggio, and Vladimir Vapnik. Feature selection for SVMs. In *Advances in Neural Information Processing Systems 13*, pages 668–674, 2000.

[3] Alain Rakotomamonjy. Variable selection using svm based criteria. *The Journal of Machine Learning Research*, 3:1357–1370, 2003.

[4] Jason Weston, André Elisseeff, Bernhard Schölkopf, and Mike Tipping. Use of the zero norm with linear models and kernel methods. *The Journal of Machine Learning Research*, 3:1439–1461, March 2003.

[5] Stephen Boyd and Lieven Vandenberghe. *Convex Optimization*. Cambridge University Press, 2004. http://www.stanford.edu/~boyd/cvxbook.html.

[6] R. Meir and T. Zhang. Generalization bounds for Bayesian mixture algorithms. *Journal of Machine Learning Research*, 4:839–860, 2003.

[7] Glenn Fung and O. L. Mangasarian. Data selection for support vector machines classifiers. In *Proceedings of the Sixth ACM SIGKDD International Conference on Knowledge Discovery and Data Mining*, pages 64–70, 2000.

[8] Simon Perkins, Kevin Lacker, and James Theiler. Grafting: Fast, incremental feature selection by gradient descent in function space. *Journal of Machine Learning Research*, 3:1333–1356, March 2003.
